# Neural Network Application to Diagnostics and Control of Vehicle Control Systems

Kenneth A. Marko
Research Staff
Ford Motor Company
Dearborn, Michigan 48121

## ABSTRACT

Diagnosis of faults in complex, real–time control systems is a complicated task that has resisted solution by traditional methods. We have shown that neural networks can be successfully employed to diagnose faults in digitally controlled powertrain systems. This paper discusses the means we use to develop the appropriate databases for training and testing in order to select the optimum network architectures and to provide reasonable estimates of the classification accuracy of these networks on new samples of data. Recent work applying neural nets to adaptive control of an active suspension system is presented.

## 1  INTRODUCTION

This paper reports on work performed on the application of artificial neural systems (ANS) techniques to the diagnosis and control of vehicle systems. Specifically, we have examined the diagnosis of common faults in powertrain systems and investigated the problem of developing an adaptive controller for an active suspension system.

In our diagnostic investigations we utilize neural networks routinely to establish the standards for diagnostic accuracy we can expect from analysis of vehicle data. Previously we have examined the use of various ANS paradigms to diagnosis of a wide range of faults in a carefully collected data set from a vehicle operated in a narrow range of speed and load. Subsequently, we have explored the classification of a data set with a more restricted set of faults, drawn from a much broader range of operating conditions. This step was taken as concern about needs for specific, real–time continuous diagnostics superseded the need to develop well–controlled, on–demand diagnostic testing. The

impetus arises from recently enacted legislation which dictates that such real–time diagnosis of powertrain systems will be required on cars sold in the U.S. by the mid–1990's. The difference between the two applications is simple: in the former studies it was presumed that an independent agent has identified that a fault is present, the root cause needs only to be identified. In the real–time problem, the diagnostic task is to detect and identify the fault as soon as it occurs. Consequently, the real–time application is more demanding. In analyzing this more difficult task, we explore some of the complications that arise in developing successful classification schemes for the virtually semi–infinite data streams that are produced in continuous operation of a vehicle fleet. The obstacles to realized applications of neural nets in this area often stem from the sophistication required of the classifier and the complexity of the problems addressed. The limited computational resources on–board vehicles will determine the scope of the diagnostic task and how implementations, such as ANS methods, will operate.

Finally, we briefly examine an extension of the ANS work to developing trainable controllers for non–linear dynamic systems such as active suspension systems. Preliminary work in this area indicates that effective controllers for non–linear plants can be developed efficiently, despite the exclusion of an accurate plant model from the training process. Although our studies were carried out in simulation, and accurate plant models were therefore available, the capability to develop controllers in the absence of such models is a significant step forward. Such controllers can be developed for existing, un-modeled hardware, and thereby reduce both the efforts required to develop control algorithms by conventional means and the time to program the real–time controllers.

## 2 NEURAL NET DIAGNOSTICS OF CONTROL SYSTEMS

Our interest in neural networks for diagnosis of faults in control systems stemmed from work on model–based diagnosis of faults in such systems, typically called plants. In the model–based approach, a model of the system under control is developed and used to predict the dynamic behavior of the system. With the system in operation, the plant performance is observed. The expected behavior and the observed behavior are compared, and if no differences are found, the plant is deemed to be operating normally. If deviations are found, the differences indicate that a fault of some sort is present (failure detection), and an analysis of the differences is used in an attempt to identify the cause (fault identification). Successful implementations (Min, 1987; Liubakka *et al*, 1988; Rizzoni *et al*, 1989) of fault detection and identification in complex systems linearized about selected operating points were put together utilizing mathematical constructs called failure detection filters. These filters are simply matrices which transform a set of observations (which become an input vector to the filter) of a plant into another vector space (the output vector or classification space). The form of these filters suggested to us that neural networks could be used to learn similar transforms and thereby avoid the tedious process of model development and validation and a priori identification of the detection filter matrix elements. We showed previously that complex signal patterns from operating internal combustion engines could be examined on a cycle by cycle basis (two revolutions of the common four–stroke engine cycle) and used to correctly identify faults present in the engine (Marko *et al*, 1989).

Typical data collected from an operating engine has been shown elsewhere (Marko *et al*, 1989). This demonstration was focussed on a production engine, limited to a small

operating range. One might suppose that a linear model–based diagnostic system could be constructed for such a task, if one wished to expend the time and effort, and therefore this exercise was not a strenuous test of the neural network approach. Additionally, our expert diagnostician could examine the data traces and accurately identify the faults. However, we demonstrated that this problem, which had eluded automated solution by other means up to that time, could easily be handled by neural network classifiers and encouraged us to proceed to more difficult problems for which efficient, rigorous procedures did not exist. We were prepared to tolerate developing empirical solutions to our more difficult problems, since we did not expect that a thorough analytic understanding would precede a demonstrated solution. The process outlined here utilized neural network analysis almost exclusively (predominantly back–propagation) on these problems. The understanding of the relationship of neural networks, the structure of the data and the training and testing of the classifiers emerged after acceptable solutions using the neural networks methods were obtained.

Consequently, the next problem addressed was that of identifying similar faults by observing the system through the multiplex serial communication link resident on the engine control computer. The serial link provides a simple hook–up procedure to the vehicle without severing any links between plant and microcontroller. However, the chief drawback of this approach is that it greatly complicates the recognition task. The complication arises because the data from the plant is sampled too infrequently, is "contaminated" by some processing in the controller, and delivered asynchronously to the serial link with respect to events in the plant (the data output process is not permitted to interrupt the real–time control requirements). In this case, a test sample of a smaller number of faults was drawn from a vehicle operated in a similar limited range to the first example and an attempt to detect and identify the faults was made using a variety of networks. Unlike the previous case, it was impossible for any experienced technicians to identify the faults. Again, neural network classifiers were found to develop satisfactory solutions over these limited data sets, which were later verified by a number of careful statistical tests (Marko *et al*, 1990). This more complex problem also produced a wider range of performance among the various neural net paradigms studied, as shown in Figure 1, where the error rates for various classifiers on these data sets are shown in the graph. These results suggested that not only would data quality and quantity need to be controlled and improved, but that the problem itself would implicitly direct us to the choice of the classifier paradigm. These issues are more thoroughly discussed elsewhere (Marko *et al*, 1990; Weiss *et al*, 1990), but the conclusion was that a complete, acceptable solution to the real scope of this problem could not be developed with our group's resources for data collection, data verification and classifier validation.

With these two experiences in mind, we could see that the first approach was an effective means of handling the failure detection and identification (FDI) problem, while the latter, although attractive from the standpoint of easy link–up to a vehicle, was for our numerical analysis, a very difficult task. It seemed that the appropriate course was to obtain reliable data, by observing the plant directly, and to perform the classification on that data. An effective scheme to accomplish this goal is to perform the classification task in the control microprocessor which has access to the direct data. Adopting this strategy, we move the diagnostics from an off–board processor to the on–board processor, and create a new set of possibilities for diagnostics.

With diagnostics contained in the controlling processor, diagnostics can be shifted from an on–demand activity, undertaken at predetermined intervals or when the vehicle operator has detected a problem, to a continuous, real–time activity. This change implies that the diagnostic algorithms will, for the most part, be evaluating a properly operating system and only infrequently be required to detect a failure and identify the cause. Additionally, the diagnostic algorithms will have to be very compact, since the current control microprocessors have very limited time and memory for calculation compared to a off–board PC. Furthermore, the classification task will need to be learned from a sample of data which is minuscule compared to the data sets that the deployed diagnostics will have to classify. This fact imposes on the training data set the requirement that it be an accurate statistical sample of the much more voluminous real–world data. This situation must prevail because we cannot anticipate the deployment of a classifier that is undergoing continuous training. A classifier capable of continuous adaptation would require more computational capability, and quite likely a supervised learning environment. The fact is, even for relatively simple diagnostics of operating engines, assembling a large, accurate training data set off–line is a considerable task. This last issue is explored in the next paragraph, but it seems to rule out early deployment of anything other than pretrained classifiers until some experience with much larger data sets from deployed diagnostic systems is obtained.

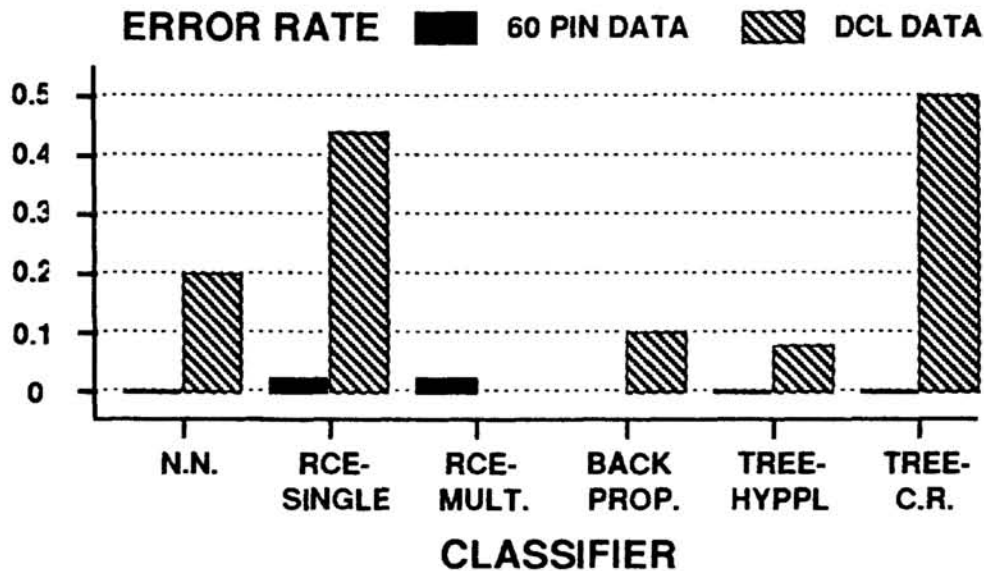

**Figure 1.** Comparison of the performance of various neural network paradigms on two static data sets by leave–one–out testing from measurements performed on vehicles in a service bay. The network paradigms tested are nearest neighbor, Restricted Coulomb Energy (RCE) Single Unit, RCE Multiple Units, Backpropagation, Tree Classifier using hyperplane separation, Tree Classifier using Center–Radius decision surface. The 60–Pin data is the data obtained directly from the engine, the DCL (Data Communication Link) data comes through the control microprocessor on a multiplexed two–wire link. Note that RCE–Multiple requires a priori knowledge about the problem which was unavailable for the DCL data and that the complete statistical testing of backpropagation was impractical due to the length of time required to train each network.

We have examined this issue of real–time diagnostics as it applies to engine misfire detection and identification. Data from normal and misfiring engines was required from a wide range of conditions, a task which consumes hours of test track driving. The set of measurements taken is extensive in order to be certain that the information obtained is a superset of the minimum set of information required. Additionally, great care needed to be exercised in establishing the accuracy of a training set for supervised learning. Specifically, we needed to be certain that the only samples of misfires included were those intentionally created, and not those which occurred spontaneously and were presumably mislabeled as normal because no intentional fault was being introduced at that time. In order to accomplish this purification of the training set, one must either have an independent detector of misfires (none exists for a production engine operating in a vehicle) or go through an iterative process to remove all the data vectors misclassified as misfire from the data set after the network has completed training. Since the independent assessment of misfire cannot be obtained, we must accept the latter method which is not altogether satisfactory. The problem with the iterative method is that one must initially exclude from the training set exactly the type of event that the system is being trained to classify. We have to start with the assumption that any additional misfires, beyond the number we introduce, are classification errors. We then reserve the right to amend this judgment in light of further experience as we build up confidence in the classifier. The results of our initial studies is shown in Fig. 2. Here we can see that a backpropagation neural network can classify a broad range of engine operation correctly, and that the network does quite well when we broaden the operating range almost to the performance limits of the engine. The classification errors indicated in the more exhaustive study are misfires detected when no misfire was introduced. At this stage of our investigation we cannot be certain that these are real errors, they may very well be misfires occurring spontaneously or appearing as a result of an additional, unintentional induced misfire in an engine cycle following the one in which the fault was introduced.

The results shown in Fig. 2 therefore represent a conservative estimate of the classification errors that can be expected from tests of our engine data. The backpropagation network we constructed demonstrated that misfire detection and identification is attainable if adequate computation resources are available and appropriate

**LIMITED OPERATION**          **EXTENDED OPERATION**

REAL CLASS

|  | NORMAL | MISFIRE |
|---|---|---|
| **NORMAL** | 762 | 0 |
| **MISFIRE** | 1 | 15 |

ANS CLASS     **NORMAL  MISFIRE**

|  | NORMAL | MISFIRE |
|---|---|---|
| **NORMAL** | 7419 | 13 |
| **MISFIRE** | 4 | 150 |

**NORMAL  MISFIRE**

**Figure 2.** Classification accuracy of a backpropagation neural network trained on misfire data tabulated as confusion matrices. Data similar to that shown in Fig. 2 is collected over a modest range of dynamic conditions and then over a very wide range of conditions (potholed roads, severe accelerations and braking etc.) to estimate the performance limits of classifiers on such data. These misclassification rates are indicators of the best possible performance obtainable from such data, and therefore they are not reasonable estimates of what practical implementations of classifiers should produce.

care in obtaining a suitable training set is exercised. However. in order to make a neural net a practical means of performing this diagnosis aboard vehicles, we need to eliminate information from the input vector which has no effect on the classification accuracy; otherwise the computational task is hopelessly beyond the capability of the engine's microcontroller. This work is currently underway using a combination of a priori knowledge about the sensor information and principal component analysis of the data sets. Nonetheless, the neural network analysis has once again established that a solution exists and set standards for classification accuracy that we can hope to emulate with more compact forms of classifiers.

## 3  NEURAL NET CONTROL OF ACTIVE SUSPENSION

The empirical approach to developing solutions for diagnostic problems suggested that a similar tactic might be employed effectively to control problems for which developing acceptable controllers for non–linear dynamic systems by conventional means was a daunting task. We wished to explore the application of feed–forward networks to the problem of learning to control a model of a non–linear active suspension system. This problem was of interest because considerable effort had gone into designing controllers by conventional means and a performance comparison could readily be made. In addition, since active suspension systems are being investigated by a number of companies, we wished to examine the possibility of developing model–independent controllers for such systems, since effective hardware systems are usually available before thoroughly validated system models appear. The initial results of this investigation, outlined below, are quite encouraging.

A backpropagation network was trained to emulate an existing controller for an active suspension as a first exercise to establish some feel for the complexity of the network required to perform such a task. A complete description of the work can be found elsewhere (Hampo, 1990), but briefly, a network with several hidden nodes was trained to provide performance equivalent to the conventional controller. Since this exercise simply replicated an existing controller, the next step was to develop a controller in the absence of any conventional controller. Therefore, a system model with a novel non–linearity was developed and utilized to train a neural network to control such a plant. The architecture for this control system is similar to that used by Nygen and Widrow (Ngyen *et al*, 1990) and is described in detail elsewhere.(Hampo *et al*, 1991) Once again, a backpropagation network, with only 2 hidden nodes, was trained to provide an satisfactory performance in controlling the suspension system simulation running on a workstation. This small network learned the task with less than 1000 training vectors, the equivalent of less than 100 feet of bumpy road.

Finally, we examined the performance of the neural network on the same plant, but without explicit use of the plant model in the control architecture. In this scheme, the output error is derived from the difference between the observed performance and the desired performance produced by a cost function based upon conventional measures of suspension performance. In this Cost Function architecture, networks of similar size were readily trained to control non–linear plants and attain performance equivalent to conventional controllers hand–tuned for such plants. Controllers developed in this manner provide a flexible means of approaching the problem of investigating tradeoffs between the conflicting demands made on such suspension systems. These demands

include ride quality, vehicle control, and energy management. This control architecture is being applied both to simulations of new systems and to actual, un-modeled hardware rigs to expedite prototype development.

# 4   CONCLUSIONS

This brief summary of our investigations has shown that neural networks play an important role in the development both of classification systems for diagnosis of faults in control systems and of controllers for practical non–linear plants. In these tasks, neural networks must compete with conventional methods. Conventional methods, although endowed with a more thorough analytic understanding, have usually failed to provide acceptable solutions to the problems we encountered as readily as have the neural network methods. Therefore, the ANS methods have a crucial role in developing solutions. Although neural networks provide these solutions expeditiously, we are just beginning to understand how these solutions arise. The growth of this understanding will determine the role neural networks play in the deployed implementations of these solutions.

## References

1. P.S. Min, "Detection of Incipient Failures in Dynamic Systems", Ph.D. Thesis, University of Michigan, 1987.

2. M.K. Liubakka, G. Rizzoni, W.B. Ribbens and K.A. Marko, "Failure Detection Algorithms Applied to Control System Design for Improved Diagnostics and Reliability", SAE Paper #880726, Detroit, Michigan, 1988.

3. G. Rizzoni, R. Hampo, M.K. Liubakka and K.A. Marko, "Real–Time Detection Filters for On–Board Diagnosis of Incipient Failures", SAE Paper #890763, 1989.

4. K.A. Marko, J. James, J. Dosdall and J. Murphy, "Automotive Control System Diagnostics Using Neural Nets for Rapid Classification of Large Data Sets", Proceedings IJCNN, II–13, Washington, D.C., 1989.

5. K.A. Marko, L.A. Feldkamp and G.V. Puskorius, "Automotive Diagnostics Using Trainable Classifiers: Statistical Testing and Paradigm Selection", Proceedings IJCNN, I–33, San Diego, California, 1990.

6. Sholom Weiss and Casimir Kulikowski, "Computer Systems That Learn", Morgan Kaufman, San Mateo, California, 1990.

7. R.J. Hampo, "Neural Net Control of an Active Suspension System", M.S. Thesis, University of Michigan, 1990.

8. D. Ngyen and B. Widrow, "The Truck–Backer Upper: An Example of Self–Learning in Neural Networks", in Neural Networks for Control, ed. W.T. Miller, MIT Press, Cambridge, Massachusetts, 1990.

9. R.J. Hampo and K.A. Marko, "Neural Net Architectures for Active Suspension Control", paper submitted to IJCNN, Seattle, Washington, 1991.
